# LEARNING REPRESENTATIONS BY RECIRCULATION

Geoffrey E. Hinton
Computer Science and Psychology Departments, University of Toronto,
Toronto M5S 1A4, Canada

James L. McClelland
Psychology and Computer Science Departments, Carnegie-Mellon University,
Pittsburgh, PA 15213

## ABSTRACT

We describe a new learning procedure for networks that contain groups of non-linear units arranged in a closed loop. The aim of the learning is to discover codes that allow the activity vectors in a "visible" group to be represented by activity vectors in a "hidden" group. One way to test whether a code is an accurate representation is to try to reconstruct the visible vector from the hidden vector. The difference between the original and the reconstructed visible vectors is called the reconstruction error, and the learning procedure aims to minimize this error. The learning procedure has two passes. On the first pass, the original visible vector is passed around the loop, and on the second pass an average of the original vector and the reconstructed vector is passed around the loop. The learning procedure changes each weight by an amount proportional to the product of the "presynaptic" activity and the *difference* in the post-synaptic activity on the two passes. This procedure is much simpler to implement than methods like back-propagation. Simulations in simple networks show that it usually converges rapidly on a good set of codes, and analysis shows that in certain restricted cases it performs gradient descent in the squared reconstruction error.

## INTRODUCTION

Supervised gradient-descent learning procedures such as back-propagation[1] have been shown to construct interesting internal representations in "hidden" units that are not part of the input or output of a connectionist network. One criticism of back-propagation is that it requires a teacher to specify the desired output vectors. It is possible to dispense with the teacher in the case of "encoder" networks[2] in which the desired output vector is identical with the input vector (see Fig. 1). The purpose of an encoder network is to learn good "codes" in the intermediate, hidden units. If for, example, there are less hidden units than input units, an encoder network will perform data-compression[3]. It is also possible to introduce other kinds of constraints on the hidden units, so we can view an encoder network as a way of ensuring that the input can be reconstructed from the activity in the hidden units whilst also making

This research was supported by contract N00014-86-K-00167 from the Office of Naval Research and a grant from the Canadian National Science and Engineering Research Council. Geoffrey Hinton is a fellow of the Canadian Institute for Advanced Research. We thank Mike Franzini, Conrad Galland and Geoffrey Goodhill for helpful discussions and help with the simulations.

the hidden units satisfy some other constraint.

A second criticism of back-propagation is that it is neurally implausible (and hard to implement in hardware) because it requires all the connections to be used backwards and it requires the units to use different input-output functions for the forward and backward passes. Recirculation is designed to overcome this second criticism in the special case of encoder networks.

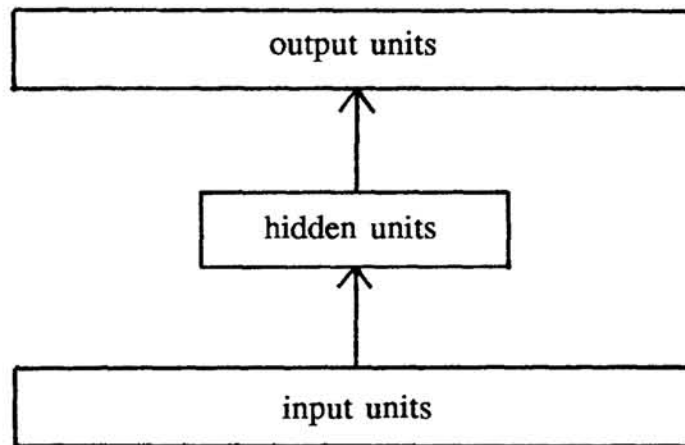

Fig. 1. A diagram of a three layer encoder network that learns good codes using back-propagation. On the forward pass, activity flows from the input units in the bottom layer to the output units in the top layer. On the backward pass, error-derivatives flow from the top layer to the bottom layer.

Instead of using a separate group of units for the input and output we use the very same group of "visible" units, so the input vector is the initial state of this group and the output vector is the state after information has passed around the loop. The difference between the activity of a visible unit before and after sending activity around the loop is the derivative of the squared reconstruction error. So, if the visible units are linear, we can perform gradient descent in the squared error by changing each of a visible unit's incoming weights by an amount proportional to the product of this difference and the activity of the hidden unit from which the connection emanates. So learning the weights from the hidden units to the output units is simple. The harder problem is to learn the weights on connections coming into hidden units because there is no direct specification of the desired states of these units. Back-propagation solves this problem by back-propagating error-derivatives from the output units to generate error-derivatives for the hidden units. Recirculation solves the problem in a quite different way that is easier to implement but much harder to analyse.

## THE RECIRCULATION PROCEDURE

We introduce the recirculation procedure by considering a very simple architecture in which there is just one group of hidden units. Each visible unit has a directed connection to every hidden unit, and each hidden unit has a directed connection to every visible unit. The total input received by a unit is

$$x_j = \sum_i y_i w_{ji} - \theta_j \tag{1}$$

where $y_i$ is the state of the $i^{\text{th}}$ unit, $w_{ji}$ is the weight on the connection from the $i^{\text{th}}$ to the $j^{\text{th}}$ unit and $\theta_j$ is the threshold of the $j^{\text{th}}$ unit. The threshold term can be eliminated by giving every unit an extra input connection whose activity level is fixed at 1. The weight on this special connection is the negative of the threshold, and it can be learned in just the same way as the other weights. This method of implementing thresholds will be assumed throughout the paper.

The functions relating inputs to outputs of visible and hidden units are smooth monotonic functions with bounded derivatives. For hidden units we use the logistic function:

$$y_j = \sigma(x_j) = \frac{1}{1+e^{-x_j}} \tag{2}$$

Other smooth monotonic functions would serve as well. For visible units, our mathematical analysis focuses on the linear case in which the output equals the total input, though in simulations we use the logistic function.

We have already given a verbal description of the learning rule for the hidden-to-visible connections. The weight, $w_{ij}$ , from the $j^{th}$ hidden unit to the $i^{th}$ visible unit is changed as follows:

$$\Delta w_{ij} = \varepsilon y_j(1) [y_i(0) - y_i(2)] \tag{3}$$

where $y_i(0)$ is the state of the $i^{th}$ visible unit at time 0 and $y_i(2)$ is its state at time 2 after activity has passed around the loop once. The rule for the visible-to-hidden connections is identical:

$$\Delta w_{ji} = \varepsilon y_i(2) [y_j(1) - y_j(3)] \tag{4}$$

where $y_j(1)$ is the state of the $j^{th}$ hidden unit at time 1 (on the first pass around the loop) and $y_j(3)$ is its state at time 3 (on the second pass around the loop). Fig. 2 shows the network exploded in time.

In general, this rule for changing the visible-to-hidden connections does not perform steepest descent in the squared reconstruction error, so it behaves differently from back-propagation. This raises two issues: Under what conditions does it work, and under what conditions does it approximate steepest descent?

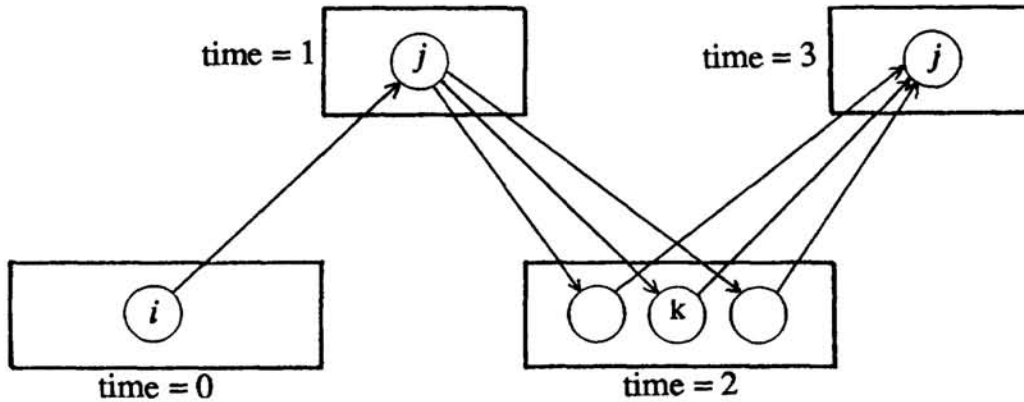

Fig. 2. A diagram showing the states of the visible and hidden units exploded in time. The visible units are at the bottom and the hidden units are at the top. Time goes from left to right.

## CONDITIONS UNDER WHICH RECIRCULATION APPROXIMATES GRADIENT DESCENT

For the simple architecture shown in Fig. 2, the recirculation learning procedure changes the visible-to-hidden weights in the direction of steepest descent in the squared reconstruction error provided the following conditions hold:

1. The visible units are linear.
2. The weights are symmetrical (i.e. $w_{ji}=w_{ij}$ for all $i,j$).
3. The visible units have high regression.

"Regression" means that, after one pass around the loop, instead of setting the activity of a visible unit, $i$, to be equal to its current total input, $x_i(2)$, as determined by Eq 1, we set its activity to be

$$y_i(2) = \lambda y_i(0) + (1-\lambda)x_i(2) \tag{5}$$

where the regression, $\lambda$, is close to 1. Using high regression ensures that the visible units only change state slightly so that when the new visible vector is sent around the loop again on the second pass, it has very similar effects to the first pass. In order to make the learning rule for the hidden units as similar as possible to the rule for the visible units, we also use regression in computing the activity of the hidden units on the second pass

$$y_j(3) = \lambda y_j(1) + (1-\lambda)\sigma(x_j(3)) \tag{6}$$

For a given input vector, the squared reconstruction error, $E$, is

$$E = \frac{1}{2}\sum_k [y_k(2)-y_k(0)]^2$$

For a hidden unit, $j$,

$$\frac{\partial E}{\partial y_j(1)} = \sum_k \frac{\partial E}{\partial y_k(2)} \frac{dy_k(2)}{dx_k(2)} \frac{\partial x_k(2)}{\partial y_j(1)} = \sum_k [y_k(2) - y_k(0)] \, y_k'(2) \, w_{kj} \qquad (7)$$

where

$$y_k'(2) = \frac{dy_k(2)}{dx_k(2)}$$

For a visible-to-hidden weight $w_{ji}$

$$\frac{\partial E}{\partial w_{ji}} = y_j'(1) \, y_i(0) \, \frac{\partial E}{\partial y_j(1)}$$

So, using Eq 7 and the assumption that $w_{kj} = w_{jk}$ for all $k, j$

$$\frac{\partial E}{\partial w_{ji}} = y_j'(1) \, y_i(0) \, [\sum_k y_k(2) \, y_k'(2) \, w_{jk} - \sum_k y_k(0) \, y_k'(2) \, w_{jk}]$$

The assumption that the visible units are linear (with a gradient of 1) means that for all $k$, $y_k'(2) = 1$. So using Eq 1 we have

$$\frac{\partial E}{\partial w_{ji}} = y_j'(1) \, y_i(0) \, [x_j(3) - x_j(1)] \qquad (8)$$

Now, with sufficiently high regression, we can assume that the states of units only change slightly with time so that

$$y_j'(1) [x_j(3) - x_j(1)] \approx \sigma(x_j(3)) - \sigma(x_j(1)) = \frac{1}{(1 - \lambda)} [y_j(3) - y_j(1)]$$

and     $y_i(0) \approx y_i(2)$

So by substituting in Eq 8 we get

$$\frac{\partial E}{\partial w_{ji}} \approx \frac{1}{(1 - \lambda)} y_i(2) [y_j(3) - y_j(1)] \qquad (9)$$

An interesting property of Eq 9 is that it does not contain a term for the gradient of the input-output function of unit $j$ so recirculation learning can be applied even when unit $j$ uses an *unknown* non-linearity. To do back-propagation it is necessary to know the gradient of the non-linearity, but recirculation *measures* the gradient by measuring the effect of a small difference in input, so the term $y_j(3) - y_j(1)$ implicitly contains the gradient.

## A SIMULATION OF RECIRCULATION

From a biological standpoint, the symmetry requirement that $w_{ij}=w_{ji}$ is unrealistic unless it can be shown that this symmetry of the weights can be learned. To investigate what would happen if symmetry was not enforced (and if the visible units used the same non-linearity as the hidden units), we applied the recirculation learning procedure to a network with 4 visible units and 2 hidden units. The visible vectors were 1000, 0100, 0010 and 0001, so the 2 hidden units had to learn 4 different codes to represent these four visible vectors. All the weights and biases in the network were started at small random values uniformly distributed in the range −0.5 to +0.5. We used regression in the hidden units, even though this is not strictly necessary, but we ignored the term $1/(1-\lambda)$ in Eq 9.

Using an $\varepsilon$ of 20 and a $\lambda$ of 0.75 for both the visible and the hidden units, the network learned to produce a reconstruction error of less than 0.1 on every unit in an average of 48 weight updates (with a maximum of 202 in 100 simulations). Each weight update was performed after trying all four training cases and the change was the sum of the four changes prescribed by Eq 3 or 4 as appropriate. The final reconstruction error was measured using a regression of 0, even though high regression was used during the learning. The learning speed is comparable with back-propagation, though a precise comparison is hard because the optimal values of $\varepsilon$ are different in the two cases. Also, the fact that we ignored the term $1/(1-\lambda)$ when modifying the visible-to-hidden weights means that recirculation tends to change the visible-to-hidden weights more slowly than the hidden-to-visible weights, and this would also help back-propagation.

It is not immediately obvious why the recirculation learning procedure works when the weights are not constrained to be symmetrical, so we compared the weight changes prescribed by the recirculation procedure with the weight changes that would cause steepest descent in the sum squared reconstruction error (i.e. the weight changes prescribed by back-propagation). As expected, recirculation and back-propagation agree on the weight changes for the hidden-to-visible connections, even though the gradient of the logistic function is not taken into account in weight adjustments under recirculation. (Conrad Galland has observed that this agreement is only slightly affected by using visible units that have the non-linear input-output function shown in Eq 2 because at any stage of the learning, all the visible units tend to have similar slopes for their input-output functions, so the non-linearity scales all the weight changes by approximately the same amount.)

For the visible-to-hidden connections, recirculation initially prescribes weight changes that are only randomly related to the direction of steepest descent, so these changes do not help to improve the performance of the system. As the learning proceeds, however, these changes come to agree with the direction of steepest descent. The crucial observation is that this agreement occurs *after* the hidden-to-visible weights have changed in such a way that they are approximately aligned (symmetrical up to a constant factor) with the visible-to-hidden weights. So it appears that changing the hidden-to-visible weights in the direction of steepest descent creates the conditions that are necessary for the recirculation procedure to cause changes in the visible-to-hidden weights that follow the direction of steepest descent.

It is not hard to see why this happens if we start with random, zero-mean visible-to-hidden weights. If the visible-to-hidden weight $w_{ji}$ is positive, hidden unit j will tend to have a higher than average activity level when the $i^{th}$ visible unit has a higher than average activity. So $y_j$ will tend to be higher than average when the reconstructed value of $y_i$ should be higher than average -- i.e. when the term $[y_i(0)-y_i(2)]$ in Eq 3 is positive. It will also be lower than average when this term is negative. These relationships will be reversed if $w_{ji}$ is negative, so $w_{ij}$ will grow faster when $w_{ji}$ is positive than it will when $w_{ji}$ is negative. Smolensky[4] presents a mathematical analysis that shows why a similar learning procedure creates symmetrical weights in a purely linear system. Williams[5] also analyses a related learning rule for linear systems which he calls the "symmetric error correction" procedure and he shows that it performs principle components analysis. In our simulations of recirculation, the visible-to-hidden weights become aligned with the corresponding hidden-to-visible weights, though the hidden-to-visible weights are generally of larger magnitude.

## A PICTURE OF RECIRCULATION

To gain more insight into the conditions under which recirculation learning produces the appropriate changes in the visible-to-hidden weights, we introduce the pictorial representation shown in Fig. 3. The initial visible vector, $A$, is mapped into the reconstructed vector, $C$, so the error vector is $AC$. Using high regression, the visible vector that is sent around the loop on the second pass is $P$, where the difference vector $AP$ is a small fraction of the error vector $AC$. If the regression is sufficiently high and all the non-linearities in the system have bounded derivatives and the weights have bounded magnitudes, the difference vectors $AP$, $BQ$, and $CR$ will be very small and we can assume that, to first order, the system behaves linearly in these difference vectors. If, for example, we moved $P$ so as to double the length of $AP$ we would also double the length of $BQ$ and $CR$.

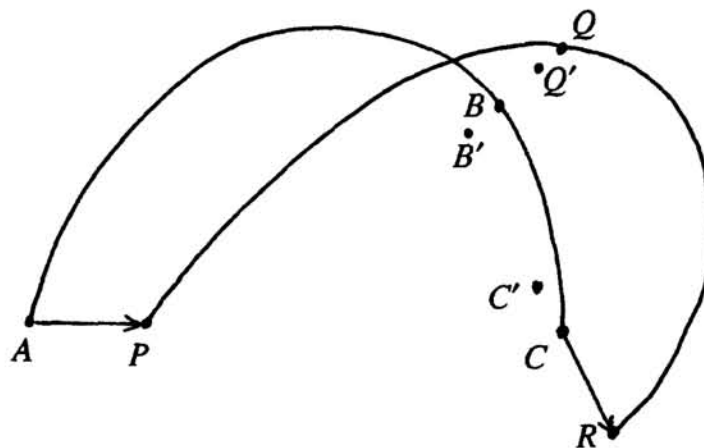

Fig. 3. A diagram showing some vectors $(A, P)$ over the visible units, their "hidden" images $(B, Q)$ over the hidden units, and their "visible" images $(C, R)$ over the visible units. The vectors $B'$ and $C'$ are the hidden and visible images of $A$ after the visible-to-hidden weights have been changed by the learning procedure.

Suppose we change the visible-to-hidden weights in the manner prescribed by Eq 4, using a very small value of ε. Let $Q'$ be the hidden image of $P$ (i.e. the image of $P$ in the hidden units) after the weight changes. To first order, $Q'$ will lie between $B$ and $Q$ on the line $BQ$. This follows from the observation that Eq 4 has the effect of moving each $y_j(3)$ towards $y_j(1)$ by an amount proportional to their difference. Since $B$ is close to $Q$, a weight change that moves the hidden image of $P$ from $Q$ to $Q'$ will move the hidden image of $A$ from $B$ to $B'$, where $B'$ lies on the extension of the line $BQ$ as shown in Fig. 3. If the hidden-to-visible weights are not changed, the visible image of $A$ will move from $C$ to $C'$, where $C'$ lies on the extension of the line $CR$ as shown in Fig. 3. So the visible-to-hidden weight changes will reduce the squared reconstruction error provided the vector $CR$ is approximately parallel to the vector $AP$.

But why should we expect the vector $CR$ to be aligned with the vector $AP$? In general we should not, except when the visible-to-hidden and hidden-to-visible weights are approximately aligned. The learning in the hidden-to-visible connections has a tendency to cause this alignment. In addition, it is easy to modify the recirculation learning procedure so as to increase the tendency for the learning in the hidden-to-visible connections to cause alignment. Eq 3 has the effect of moving the visible image of $A$ closer to $A$ by an amount proportional to the magnitude of the error vector $AC$. If we apply the same rule on the next pass around the loop, we move the visible image of $P$ closer to $P$ by an amount proportional to the magnitude of $PR$. If the vector $CR$ is anti-aligned with the vector $AP$, the magnitude of $AC$ will exceed the magnitude of $PR$, so the result of these two movements will be to improve the alignment between $AP$ and $CR$. We have not yet tested this modified procedure through simulations, however.

This is only an informal argument and much work remains to be done in establishing the precise conditions under which the recirculation learning procedure approximates steepest descent. The informal argument applies equally well to systems that contain longer loops which have several groups of hidden units arranged in series. At each stage in the loop, the same learning procedure can be applied, and the weight changes will approximate gradient descent provided the difference of the two visible vectors that are sent around the loop aligns with the difference of their images. We have not yet done enough simulations to develop a clear picture of the conditions under which the changes in the hidden-to-visible weights produce the required alignment.

## USING A HIERARCHY OF CLOSED LOOPS

Instead of using a single loop that contains many hidden layers in series, it is possible to use a more modular system. Each module consists of one "visible" group and one "hidden" group connected in a closed loop, but the visible group for one module is actually composed of the hidden groups of several lower level modules, as shown in Fig. 4. Since the same learning rule is used for both visible and hidden units, there is no problem in applying it to systems in which some units are the visible units of one module and the hidden units of another. Ballard[6] has experimented with back-propagation in this kind of system, and we have run some simulations of recirculation using the architecture shown in Fig. 4. The network

learned to encode a set of vectors specified over the bottom layer. After learning, each of the vectors became an attractor and the network was capable of completing a partial vector, even though this involved passing information through several layers.

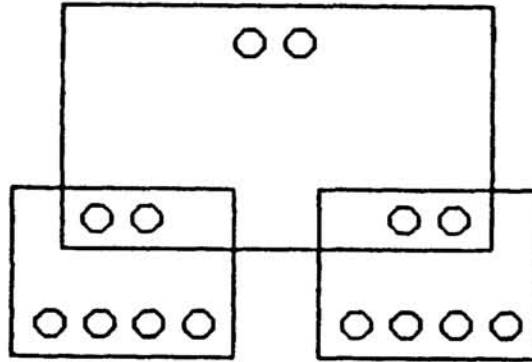

Fig 4. A network in which the hidden units of the bottom two modules are the visible units of the top module.

## CONCLUSION

We have described a simple learning procedure that is capable of forming representations in non-linear hidden units whose input-output functions have bounded derivatives. The procedure is easy to implement in hardware, even if the non-linearity is unknown. Given some strong assumptions, the procedure performs gradient descent in the reconstruction error. If the symmetry assumption is violated, the learning procedure still works because the changes in the hidden-to-visible weights produce symmetry. If the assumption about the linearity of the visible units is violated, the procedure still works in the cases we have simulated. For the general case of a loop with many non-linear stages, we have an informal picture of a condition that must hold for the procedure to approximate gradient descent, but we do not have a formal analysis, and we do not have sufficient experience with simulations to give an empirical description of the general conditions under which the learning procedure works.

## REFERENCES

1. D. E. Rumelhart, G. E. Hinton and R. J. Williams, *Nature* **323**, 533-536 (1986).

2. D. H. Ackley, G. E. Hinton and T. J. Sejnowski, *Cognitive Science* **9**,147-169 (1985).

3. G. Cottrell, J. L. Elman and D. Zipser, Proc. Cognitive Science Society, Seattle, WA (1987).

4. P. Smolensky, Technical Report CU-CS-355-87, University of Colorado at Boulder (1986).

5. R. J. Williams, Technical Report 8501, Institute of Cognitive Science, University of California, San Diego (1985).

6. D. H. Ballard, Proc. American Association for Artificial Intelligence, Seattle, WA (1987).
